# Extending position/phase-shift tuning to motion energy neurons improves velocity discrimination

**Stanley Yiu Man Lam and Bertram E. Shi**
Department of Electronic and Computer Engineering
Hong Kong Univeristy of Science and Technology
Clear Water Bay, Kowloon, Hong Kong
{eelym,eebert}@ee.ust.hk

## Abstract

We extend position and phase-shift tuning, concepts already well established in the disparity energy neuron literature, to motion energy neurons. We show that Reichardt-like detectors can be considered examples of position tuning, and that motion energy filters whose complex valued spatio-temporal receptive fields are space-time separable can be considered examples of phase tuning. By combining these two types of detectors, we obtain an architecture for constructing motion energy neurons whose center frequencies can be adjusted by both phase and position shifts. Similar to recently described neurons in the primary visual cortex, these new motion energy neurons exhibit tuning that is between purely space-time separable and purely speed tuned. We propose a functional role for this intermediate level of tuning by demonstrating that comparisons between pairs of these motion energy neurons can reliably discriminate between inputs whose velocities lie above or below a given reference velocity.

## 1 Introduction

Image motion is an important cue used by both biological and artificial visual systems to extract information about the environment. Although image motion is commonly used, there are different models for image motion processing in different systems. The Reichardt model is a dominant model for motion detection in insects, where image motion analysis occurs at a very early stage [1]. For mammals, the bulk of visual processing for motion is thought to occur in the cortex, and the motion energy model is one of the dominant models [2][3]. However, despite the differences in complexity between these two models, they are mathematically equivalent given appropriate choices of the spatial and temporal filters [4].

The motion energy model is very closely related to the disparity energy model, which has been used to model the outputs of disparity selective neurons in the visual cortex [5]. The disparity tuning of neurons in this model can be adjusted via two mechanisms: a position shift between the center locations of the receptive fields in the left and right eyes or a phase shift between the receptive field organization in the left and right eyes [6][7]. It appears that biological systems use a combination of these two mechanisms.

In Section 2, we extend the concepts of position and phase tuning to the construction of motion energy neurons. We combine the Reichardt model and the motion energy model to obtain an architecture for constructing motion energy neurons whose tuning can be adjusted by the analogs of position and phase shifts. In Section 3, we investigate the functional advantages of position and phase shifts, inspired by a similar comparison from the disparity energy literature. We show that by simply comparing the outputs of pair of motion energy cells with combined position/phase shift tuning enables us to discriminate reliably between stimuli moving above and below a reference

velocity. Finally, in Section 4, we place this work in the context of recent results on speed tuning in V1 neurons.

## 2 Extending Position/Phase Tuning to Motion Energy Models

Figure 1(a) shows a 1D array of three Reichardt detectors[1] tuned to motion from left to right. Each detector computes the correlation between its photosensor input and the delayed input from the photosensor to the left. The delay could be implemented by a low pass filter. Usually, the correlation is assumed to be computed by a multiplication between the current and delayed signals. For consistency with the following discussion, we show the output as a summation followed by a squaring. Squaring the sum is essentially equivalent to the product, since the product could be recovered by subtracting the sum of the squared inputs from the squared sum (e.g. $(a + b)^2 - (a^2 + b^2) = 2ab$).

Delbruck proposed a modification of the Reichardt detector (Figure 1(b)), which switches the order of the delay and the sum, resulting in a delay-line architecture [8]. The output of a detector is the sum of its photosensor input and the delayed output of the detector to the left. This recurrent connection extends the spatio-temporal receptive field of the detector, since the input from the second-nearest-neighboring photosensor to the left is now connected to the detector through two delays, whereas the Reichardt detector never sees the output of its second-nearest-neighboring photosensor.

The velocity tuning of these detectors is determined by the combination of the temporal delay and the position shift between the neighboring detectors. As the delay increases, the tuned velocity decreases. As the position shift increases, the tuned velocity also increases. This position-tuning of velocity is reminiscent of the position-tuning of disparity energy neurons, where the larger the position shift between the spatial receptive fields being combined from the left and right eyes, the larger the disparity tuning [9].

Figure 1(c) shows a 1D array of three motion energy detectors[2][3]. At each spatial location, the outputs of the photosensors in a neighborhood around each spatial location are combined through even and odd symmetric linear spatial receptive fields, which are here modelled by spatial Gabor functions. In 1D, the even and odd symmetric Gabor receptive field profiles are the real and imaginary parts of the function

$$g_s(x) = \frac{1}{\sqrt{2\pi}\sigma_x} \exp\left(-\frac{x^2}{2\sigma_x^2}\right) \exp(jx\Omega_x) = \frac{1}{\sqrt{2\pi}\sigma_x} \exp\left(-\frac{x^2}{2\sigma_x^2}\right) (\cos(\Omega_x x) + j\sin(\Omega_x x)) \quad (1)$$

where $\Omega_x$ determines the preferred spatial frequency of the receptive field, and $\sigma_x$ determines its spatial extent. The even and odd spatial filter outputs are then combined through temporal filters to produce two outputs which are then squared and summed to produce the motion energy. In many cases, the temporal receptive field profiles are also Gabor functions. The combined spatial and temporal receptive fields of the two neurons are separable when considered as a single complex valued function:

$$g(x, t) = \frac{1}{\sqrt{2\pi}\sigma_x} \exp\left(-\frac{x^2}{2\sigma_x^2}\right) \exp(j\Omega_x x) \cdot \frac{1}{\sqrt{2\pi}\sigma_t} \exp\left(-\frac{t^2}{2\sigma_t^2}\right) \exp(j\Omega_t t) \quad (2)$$

where $\Omega_t$ and $\sigma_t$ determine the preferred temporal frequency and temporal extent of the temporal receptive fields. Strictly speaking, these spatio-temporal filters are not velocity tuned, since the velocity at which a moving sine-wave grating stimulus produces maximum response varies with the spatial frequency of the sine-wave grating. However, since spatial frequencies of $\Omega_x$ lead to the largest responses, the filter is sometimes thought of as having a preferred velocity $V_{\text{pref}} = -\Omega_t/\Omega_x$.

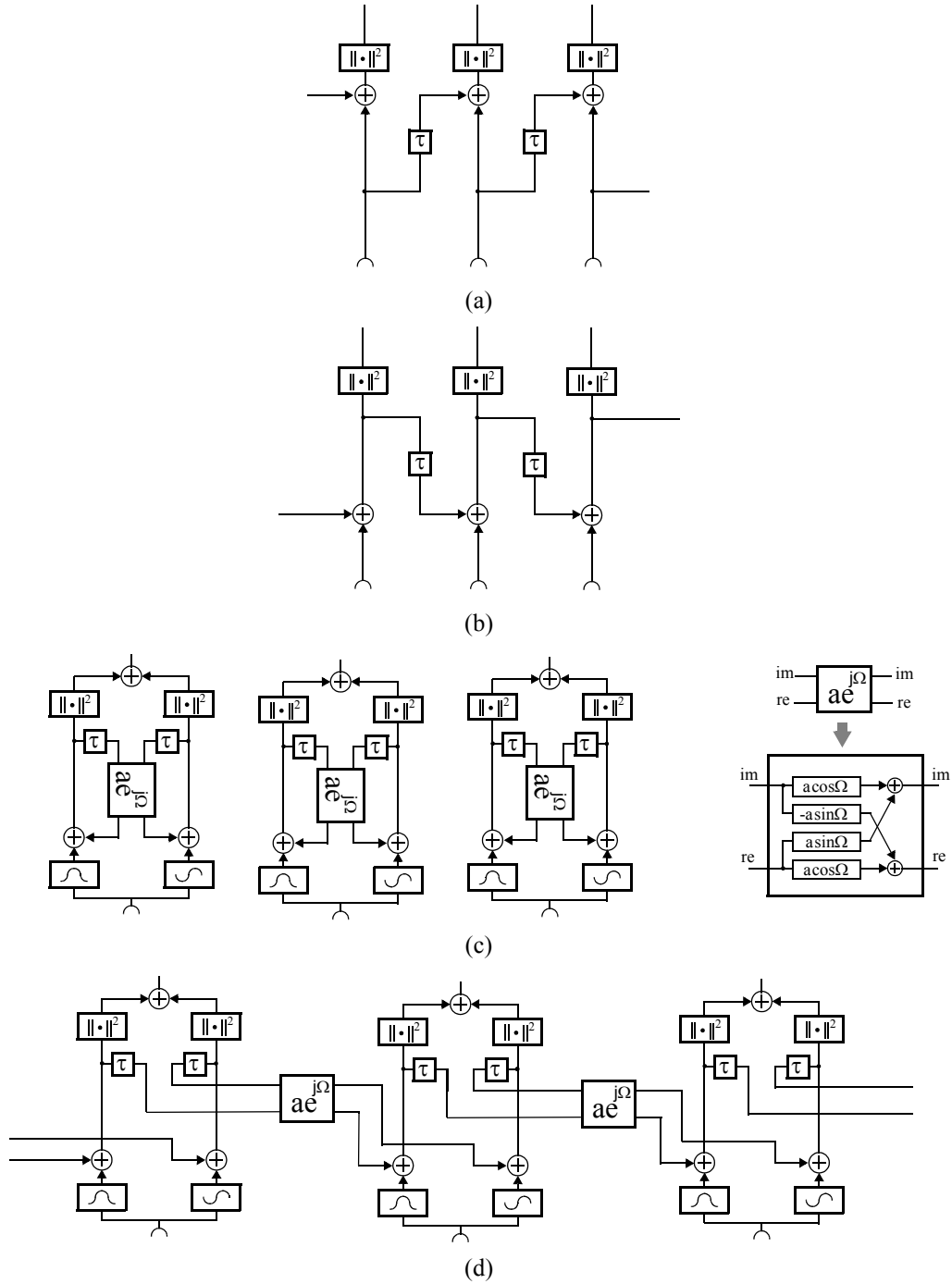

**Figure 1.** (a) 1D array of three Reichardt detectors tuned to motion from left to right. The $\tau$ block represents a temporal delay. The semi-circles represent photosensors. (b) Delbruck delay-line detector. (c) 1D array of three motion energy detectors. The bottom blocks represent even and odd symmetric spatial receptive fields modelled by Gabor functions. (d) The proposed motion detector by combining the position and phase tuning mechanisms of (b) and (c).

One problem with using spatio-temporal Gabor functions is that they are non-causal in time. In this work, we consider the use of a causal recurrently implemented temporal filter. If we let the real and imaginary parts of $u(x, t)$ denote the even and odd spatial filter outputs, then the two temporal fil-

ter outputs of the temporal filter are given by the real and imaginary parts of $v(x, t)$, which satisfies

$$v(x, t) = a \exp(j\Omega_t) \cdot v(x, t-1) + (1-a) \cdot u(x, t) \qquad (3)$$

where $a < 1$ and $\Omega_t$ are real valued constants. We derive this equation from Fig. 1(c) by considering the time delay $\tau$ as a unit sample discrete time delay. We consider discrete time operation here for consistency with our experimental results, however, a corresponding continuous time temporal filter can be obtained by replacing the time delay by a first order continuous-time recurrent filter with time constant $\tau$. The frequency response of this complex-valued filter is

$$\frac{V(\omega_x, \omega_t)}{U(\omega_x, \omega_t)} = \frac{1-a}{1 - a \cdot \exp(-j(\omega_t - \Omega_t))} \qquad (4)$$

where $\omega_x$ and $\omega_t$ are spatial and temporal frequency variables. This function achieves unity maximum value at $\omega_t = -\Omega_t$, independently of $\omega_x$. Assuming the same Gabor spatial receptive field, the combined spatio-temporal receptive field can be approximated by the continuous function:

$$g(x, t) = \frac{1}{\sqrt{2\pi}\sigma_x} \exp\left(-\frac{x^2}{2\sigma_x^2}\right) \exp(j\Omega_x x) \cdot \tau^{-1} \exp(-t/\tau) \exp(j\Omega_t t) h(t) \qquad (5)$$

where $h(t)$ is the unit step function, and $\tau^{-1} \approx (1-a)$. Again, strictly speaking, the filter is not velocity tuned, but for input sine-wave gratings with a spatial frequency near $\Omega_x$, the composite spatio-temporal filter has a preferred velocity near $v_{\mathrm{pref}} = -\Omega_t/\Omega_x$.

The velocity tuning of this filter is determined by the combination of the time delay and a phase shift $\Omega_t$ between the input $u(x, t)$ and the output $v(x, t-1)$. The longer the time delay, the slower the preferred velocity. However, the larger the phase-shift, the higher the preferred velocity. This phase-tuning of velocity is reminiscent of the phase-tuning of disparity tuned neurons, where the larger the phase shift between the left and right receptive fields, the larger the preferred disparity.

The possibility to adjust velocity tuning using two complementary mechanisms, suggests that it should be possible to combine these two methods, as observed in disparity neurons. Figure 1(d) shows how the position and phase tuning mechanisms of Figures 1(b) and 1(c) can be combined. The preferred velocity for spatial frequencies $\Omega_x$ will be determined by the sum of the preferred velocities determined by the position and phase-shift mechanisms, i.e. $v_{\mathrm{pref}} = 1 - \Omega_t/\Omega_x$, assuming a unit spatial displacement between adjacent photosensors.

## 3 Motion energy pairs for velocity discrimination

Given the possibility of combining the position and phase tuning mechanisms, an interesting question is how these two mechanisms might be exploited when constructing populations of motion energy neurons. Velocity can be estimated using a population of neurons tuned to different spatio-temporal frequencies [10][11]. However, the output of a single motion energy neuron is an ambiguous indicator of velocity, since its output depends upon other stimulus dimensions in addition to motion, (e.g. orientation, contrast).

Given the long history of position/phase shifts in disparity tuning, it is natural to start with an inspiration taken from the context of binocular vision. It has been shown that the responses from a population of phase-tuned disparity energy are more comparable than the responses from a population of position-tuned disparity energy neurons [12]. In particular, the preferred disparity of the neuron with maximum response in a population of phase tuned neurons is a more reliable indicator of the stimulus disparity than the preferred disparity of the neuron with maximum response in a population of position tuned neurons, especially for neurons with small phase shifts. The disadvantage of purely phase tuned neurons is that their preferred disparities can be tuned only over a limited range due to phase-wraparound in the sinusoidal modulation of the spatial Gabor. However, there is no

restriction on the range of preferred disparities when using position shifts. Thus, it has been suggested that position shifts can be used to "bias" the preferred disparity of a population around a rough estimate of the stimulus disparity, and then use a population of neurons tuned by phase shifts to obtain a more accurate estimation of the actual disparity.

In this section, we demonstrate that a similar phenomenon holds for motion energy neurons. In particular, we show that we can use position shifts to place the tuned velocity (for a spatial frequency of $\Omega_x$) in a population of two neurons around a desired bias velocity, $v_{\text{bias}}$, and then use phase shifts with equal magnitude but opposite sign to place the preferred velocities symmetrically around this bias velocity. We then show that by comparing the outputs of these two neurons, we can accurately discriminate between velocities above and below $v_{\text{bias}}$.

The equation describing the complex valued output of the spatio-temporal filtering stage $w(x, t)$ for the detector shown in Figure 1(d) is

$$w(x, t) = a \exp(j\Omega_t) \cdot w(x-1, t-1) + (1 - a) \cdot u(x, t) \tag{6}$$

The frequency response is

$$\frac{W(\omega_x, \omega_t)}{U(\omega_x, \omega_t)} = \frac{1 - a}{1 - a \cdot \exp(-j(\omega_t + \omega_x - \Omega_t))} \tag{7}$$

and achieves its maximum along the line $\omega_t = \omega_x + \Omega_t$, as seen in the contour plot of the spatio-temporal frequency response magnitude of the cascade of (1) and (7) in Fig. 2(a). In comparison, the spatio-temporal frequency response of the cascade of (1) and (4) shown in Fig. 2(e), achieves its maximum at $\Omega_t$ independently of $\omega_x$. For a moving sine wave grating input with spatial and temporal frequencies $\omega_x$ and $\omega_t$, the steady state motion energy outputs will be proportional to the squared magnitudes of the spatio-temporal frequency response evaluated at $(\omega_x, \omega_t)$.

Assume that we have two such motion cells with the same preferred spatial frequency $\Omega_x = 2\pi/20$ but opposite temporal frequencies $\Omega_t = \pm 2\pi/20$. The motion energy cell with positive $\Omega_t$ is tuned to fast velocities, while the motion energy cell with negative $\Omega_t$ is tuned to slow velocities. If we compare the frequency response magnitudes at frequency $(\omega_x, \omega_t)$, the boundary between the regions in the $\omega_x - \omega_t$ plane where the magnitude of one is larger than the other is a line passing thorough the origin with slope equal to 1, as shown in Fig. 2(c). This suggests that we can determine whether the velocity of the grating is faster or slower than 1 pixel per frame by checking the relative magnitude of the motion energy outputs, at least for sine-wave gratings.

Although the sine-wave grating is a particularly simple input, this property is not shared by other pairs of motion energy neurons. For example, Fig. 2(f) shows the spatio-temporal frequency responses two motion energy neurons that have the same spatio-temporal center frequencies as considered above, but are constructed by phase tuning (the cascade of (1) and (4)). In this case, the boundary is a horizontal line. Thus, the velocity boundary depends upon the spatial frequency. For lower spatial frequencies, the relative magnitudes will switch at higher velocities. Another commonly considered arrangement of Gabor-filters is to place the center frequencies around a circle. For two neurons, this corresponds to displacing the two center frequencies by an equal amount perpendicularly to the line $\omega_x = \omega_t$ (Fig. 2(k)). For motion energy filters built from non-causal Gabor filters, the spatio-temporal frequency responses exhibit perfect circular symmetry, and the decision boundary also coincides with the diagonal line $\omega_x = \omega_t$ (see Figure 9 in [13]). However, non-causal filters are not physically realizable. If we consider motion energy neurons constructed from temporally causal functions (e.g. the cascade (1) and (4)), the boundary only matches the diagonal line in a small neighborhood of $\omega_x = \Omega_x$, as shown in Fig. 2(i).

We have characterized the performance of the three motion pairs on the fast/slow velocity discrimination task for a variety of inputs, including sine-wave gratings, square wave gratings, and drifting random dot stimuli with varying coherence.

We first consider drifting sinusoidal gratings with spatial frequencies $\omega_x \in [0, 2\pi/10]$ and velocities $v_{\text{input}} \in [0, 2]$. For each spatial frequency and velocity, we compare the two motion energy

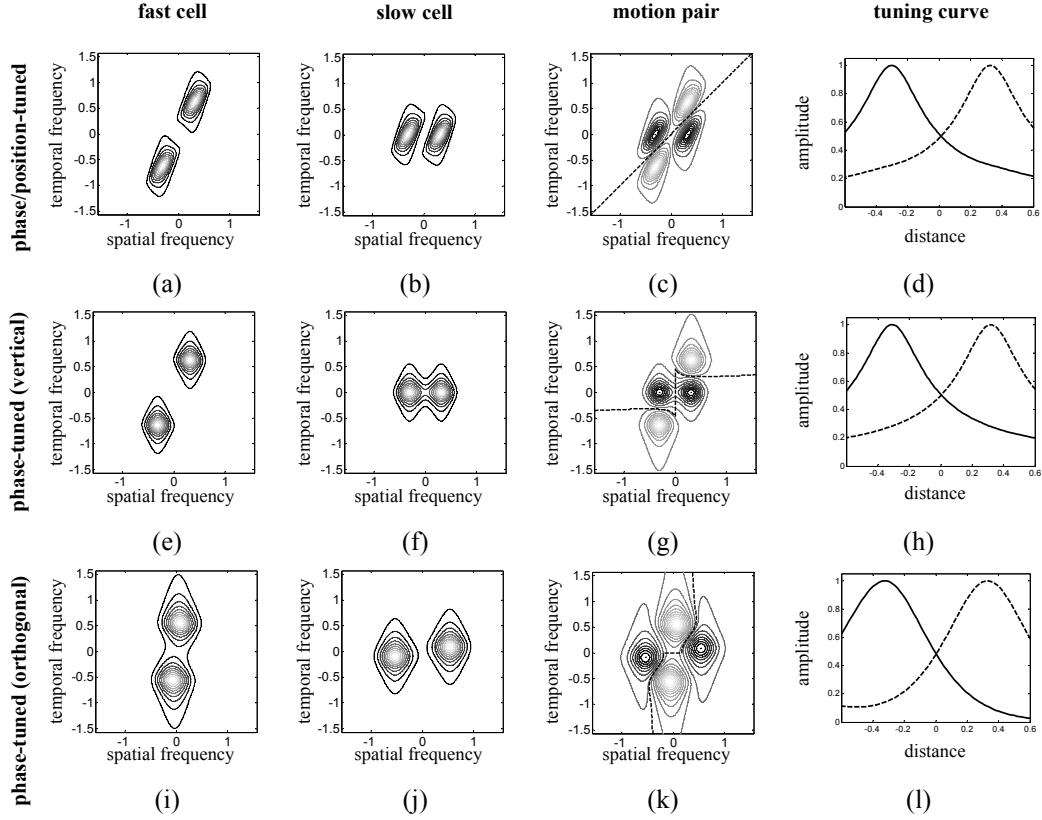

**Figure 2.** Frequency response amplitudes of the motion pairs formed by types of motion cells. First row: Phase and position tuned motion cells. The center frequencies of the fast (a) and slow (b) cells are $(\omega_x, \omega_t) = (0.314, 0.628)$ and $(0.314, 0)$ respectively. Second row: Vertically displaced phase-tuned motion energy cells. The center frequencies of the fast (e) and slow (f) cells are $(0.314, 0.628)$ and $(0.314, 0)$ respectively. Third row: Orthogonally displaced phase-tuned motion energy cells. The center frequencies of the fast (i) and slow (j) cells are $(0.092, 0.536)$ and $(0.536, 0.092)$ respectively. The third column shows the contour plot of difference between the frequency response amplitudes of the fast cell from the slow cell. The dashed line shows the decision boundary at zero. The fourth column shows the cross sections of the frequency response amplitudes along the line connecting the two center frequencies (fast = solid, slow = dashed). Zero denotes the point on the line that crosses $\omega_t = \omega_x$.

outputs at different phase shifts of the input grating, and calculate the percentage where the response of the fast cell is larger than that of the slow cell. Fig. 3(a)-(c) show the percentages as the grey scale value for each combination of input spatial frequency and velocity. Ideally, the top half should be white (i.e. the fast cell's response is larger for all inputs whose velocity is greater than one), and the bottom half should be black. For the phase-shifted motion cells with unit position-tuned velocity bias, the responses are correct over a wide range of spatial frequencies. On the other hand, for the motion pairs with the same center frequencies but tuned by pure phase shifts (Fig. 3(c)), the velocity at which the relative responses switch decreases with spatial frequency. This is consistent with the horizontal decision boundary computed by comparing the frequency response magnitudes. For the phase-tuned motion-energy cells with orthogonally displaced center frequencies, the boundary rapidly diverges from the horizontal as the spatial frequency moves away from $\Omega_x$. Fig. 3(d) shows the overall accuracy by combining the responses over all velocities. The detector utilizing the phase-tuned cells with position bias have the highest accuracy over the widest range of spatial frequencies.

Fig. 3(e)-(h) show the responses of the motion pairs to square wave gratings. The results are similar to the case of sinusoidal gratings, except that the performance at low spatial frequencies is worse.

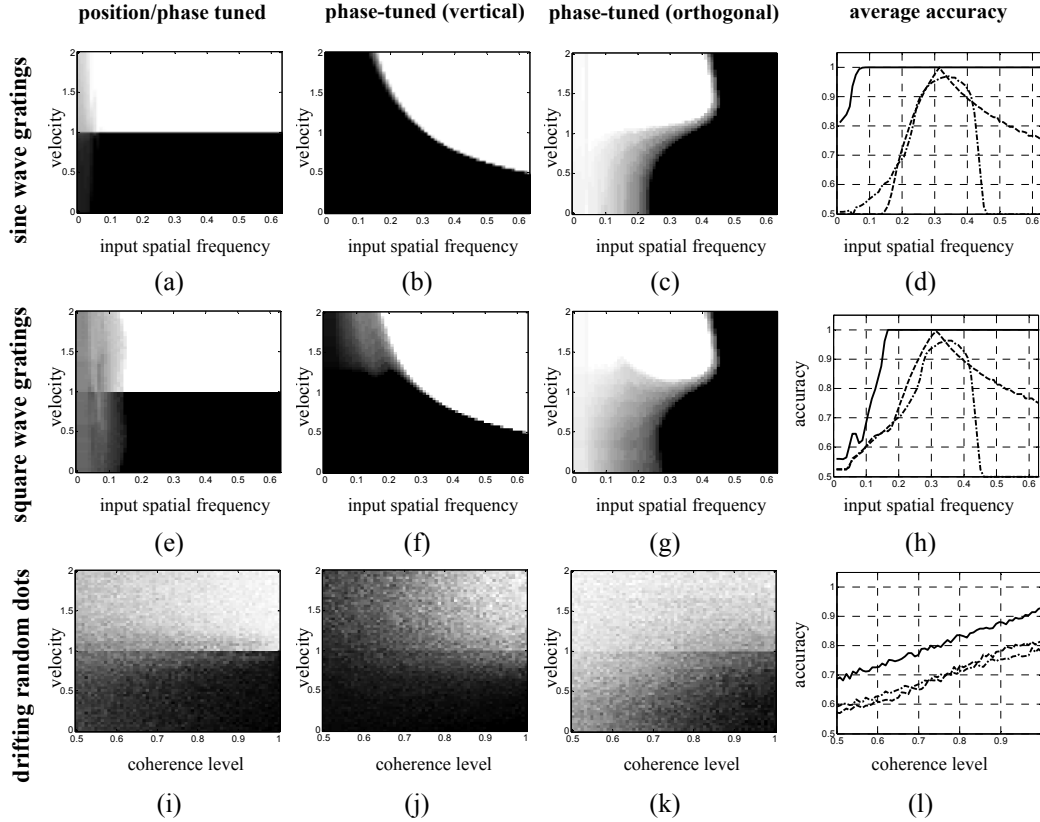

**Figure 3.** Performance on the velocity discrimination task for different stimuli. First row: sine wave gratings; second row: square wave gratings; third row: drifting random dots. The first three columns show the percentage of stimuli where the fast motion energy cell's response is larger than the slow cell's response. First column: motion cells with position-tuned velocity bias; second column: phase tuned motion cells with the same center frequencies; third column: phase-tuned motion cells with orthogonal offset. The fourth column shows the average accuracy over all input velocities. Solid line: motion cells with position-tuned velocity bias; dashed line: phase tuned motion cells with the same center frequencies; dash-dot line: phase-tuned motion cells with orthogonal offset.

this is expected, since for low spatial frequencies, the square wave gratings have large constant intensity areas that convey no motion information.

Fig. 3(i)-(l) show the responses for drifting random dot stimuli at different velocities and coherence levels. The dots were one pixel wide. The motion pair using the phase-shifted cells with position tuned bias velocity maintain a consistently higher accuracy over all coherence levels tested.

## 4 Discussion

We described a new architecture for motion energy filters obtained by combining the position tuning mechanism of the Reichardt-like detectors and the phase tuning mechanism of motion energy detectors based on complex-valued spatio-temporal separable filters. Motivated by results with disparity energy neurons indicating that the responses of phase-tuned neurons with small phase shifts are more comparable, we have examined the ability of the proposed velocity detectors to discriminate between input stimuli above and below a fixed velocity. Our experimental and analytical results confirm that comparisons between pairs constructed by using a position shift to center the tuned velocities around the border and using phase shifts to offset the tuned velocity of the pair to opposite sides of the boundary is consistently better than previously proposed architectures that were based on pure phase tuning.

Recent experimental evidence has cast doubt upon the belief that the motion neurons in V1 and MT have very distinct properties. Traditionally, the tuning of V1 motion sensitive neurons is thought to be separable along the spatial and temporal frequency dimensions, while the frequency tuning MT neurons is inseparable, consistent with constant speed tuning. However, it now seems that both V1 and MT neurons actually show a continuum in the degree to which preferred velocity changes with spatial frequency [14][15][16]. Our proposed neurons constructed by position and phase shifts also show an intermediate behavior between speed tuning and space-time separable tuning. With pure phase shifts, the tuning is space-time separable. With position shifts, the neurons become speed tuned. An intermediate tuning is obtained by combining position and phase tuning. Our results on a simple velocity discrimination task suggest a functional role for this intermediate level of tuning in creating motion energy pairs whose relative responses truly indicate changes in velocity around a reference level for stimuli with a broad band of spatial frequency content. Pair-wise comparisons have been previously proposed as a potential method for coding image speed [17][18]. Here, we have demonstrated a systematic way of constructing reliably comparable pairs of neurons using simple neurally plausible circuits.

## Acknowledgements

This work was supported in part by the Hong Kong Research Grants Council under Grant HKUST6300/04E.

## References

[1]     W. Reichardt, "Autocorrelation, a principle for the evaluation of sensory information by the central nervous system," in *Sensory Communication*, W. A. Rosenblith, ed. (Wiley, New York, 1961).

[2]     E. Adelson and J. Bergen, "Spatiotemporal energy models for the perception of motion," *Optical Society of America, Journal, A: Optics and Image Science*, vol. 2, pp. 284-299, 1985.

[3]     A. B. Watson and J. A. J. Ahumada, "Model of human visual-motion sensing," *Journal Optical Society of America A*, vol. 2, pp. 322-342, 1985.

[4]     J. P. H. van Santen and G. Sperling, "Elaborated Reichardt detectors," *Journal of the Optical Society of America A,* vol. 2, pp. 300-321, 1985.

[5]     I. Ohzawa, G. C. DeAngelis, and R. D. Freeman, "Stereoscopic depth discrimination in the visual cortex: Neurons ideally suited as disparity detectors," *Science*, vol. 249, pp. 1037-1041, 1990.

[6]     N. Qian, "Computing stereo disparity and motion with known binocular cell properties," *Neural Computation,* vol. 6, pp. 390-404, 1994.

[7]     D. Fleet, H. Wagner, and D. Heeger, "Neural encoding of binocular disparity: Energy models, position shifts and phase shifts," *Vision Research,* vol. 36, pp. 1839-1857, 1996.

[8]     T. Delbruck, "Silicon Retina with Correlation-Based, Velocity-Tuned Pixels," *IEEE Transactions on Neural Networks*, vol. 4, pp. 529-541, 1993.

[9]     A. Anzai, I. Ohzawa and R. D. Freeman, "Neural mechanisms for encoding binocular disparity: Position vs. phase," *J. Neurophysiology*, vol. 82, pp. 874-890, 1999.

[10]    D. J. Heeger, "Model for the extraction of image flow," *Journal Optical Society of America A,* vol. 4, pp. 1455-1471, 1987.

[11]    E. Simoncelli and D. Heeger, "A model of neuronal responses in visual area MT," *Vision Research*, vol. 38, pp. 743-61, 1998.

[12]    Y. Chen and N. Qian, "A course-to-fine disparity energy model with both phase-shift and position-shift receptive field mechanisms," *Neural Computation*, vol. 16, pp. 1545-1578, 2004.

[13]    M. V. Srinivasan, M. Poteser and K. Kral, "Motion detection in insect orientation and navigation," *Vision Research*, vol. 39, pp. 2749-2766, 1999.

[14]    N. Priebe, C. Cassanello, and S. Lisberger, "The Neural Representation of Speed in Macaque Area MT/V5," *Journal of Neuroscience*, vol. 23, pp. 5650, 2003.

[15]    N. Priebe, S. Lisberger, and J. Movshon, "Tuning for Spatiotemporal Frequency and Speed in Directionally Selective Neurons of Macaque Striate Cortex," *Journal of Neuroscience*, vol. 26, pp. 2941-2950, 2006.

[16]    J. Perrone, "A Single Mechanism Can Explain the Speed Tuning Properties of MT and V1 Complex Neurons," *Journal of Neuroscience*, vol. 26, pp. 11987-11991, 2006.

[17]    P. Thompson, "Discrimination of moving gratings at and above detection threshold," *Vision Research*, vol. 23, pp. 1533-1538, 1983.

[18]    J. A. Perrone, "Simulating the speed and direction tuning of MT neurons using spatiotemporal tuned V1-neuron inputs". *Investigative Opthalmology and Visual Science (Supplement)*, vol. 35, pp. 2158, 1994.
